# Coulomb Classifiers: Generalizing Support Vector Machines via an Analogy to Electrostatic Systems

**Sepp Hochreiter[†], Michael C. Mozer[∗], and Klaus Obermayer[†]**
[†]Department of Electrical Engineering and Computer Science
Technische Universität Berlin, 10587 Berlin, Germany
[∗]Department of Computer Science
University of Colorado, Boulder, CO 80309–0430, USA
{hochreit,oby}@cs.tu-berlin.de, mozer@cs.colorado.edu

## Abstract

We introduce a family of classifiers based on a physical analogy to an electrostatic system of charged conductors. The family, called *Coulomb classifiers*, includes the two best-known support-vector machines (SVMs), the $\nu$–SVM and the $C$–SVM. In the electrostatics analogy, a training example corresponds to a charged conductor at a given location in space, the classification function corresponds to the electrostatic potential function, and the training objective function corresponds to the Coulomb energy. The electrostatic framework provides not only a novel interpretation of existing algorithms and their interrelationships, but it suggests a variety of new methods for SVMs including kernels that bridge the gap between polynomial and radial-basis functions, objective functions that do not require positive-definite kernels, regularization techniques that allow for the construction of an optimal classifier in Minkowski space. Based on the framework, we propose novel SVMs and perform simulation studies to show that they are comparable or superior to standard SVMs. The experiments include classification tasks on data which are represented in terms of their pairwise proximities, where a Coulomb Classifier outperformed standard SVMs.

## 1 Introduction

Recently, Support Vector Machines (SVMs) [2, 11, 9] have attracted much interest in the machine-learning community and are considered state of the art for classification and regression problems. One appealing property of SVMs is that they are based on a convex optimization problem, which means that a single minimum exists and can be computed efficiently. In this paper, we present a new derivation of SVMs by analogy to an electrostatic system of charged conductors. The electrostatic framework not only provides a physical interpretation of SVMs, but it also gives insight into some of the seemingly arbitrary aspects of SVMs (e.g., the diagonal of the quadratic form), and it allows us to derive novel SVM approaches. Although we

are the first to make the analogy between SVMs and electrostatic systems, previous researchers have used electrostatic nonlinearities in pattern recognition [1] and a mechanical interpretation of SVMs was introduced in [9].

In this paper, we focus on the classification of an *input vector* $\boldsymbol{x} \in \mathcal{X}$ into one of two categories, labeled "+" and "−". We assume a supervised learning paradigm in which $N$ training examples are available, each example $i$ consisting of an input $\boldsymbol{x}^i$ and a label $y_i \in \{-1, +1\}$. We will introduce three electrostatic models that are directly analogous to existing machine-learning (ML) classifiers, each of which builds on and generalizes the previous. For each model, we describe the physical system upon which it is based and show its correspondence to an ML classifier.

## 1.1 Electrostatic model 1: Uncoupled point charges

Consider an electrostatic system of point charges populating a space $\mathcal{X}'$ homologous to $\mathcal{X}$. Each point charge corresponds to a particular training example; point charge $i$ is fixed at location $\boldsymbol{x}^i$ in $\mathcal{X}'$, and has a charge of sign $y_i$. We define two sets of fixed charges: $S^+ = \{\boldsymbol{x}^i \mid y_i = +1\}$ and $S^- = \{\boldsymbol{x}^i \mid y_i = -1\}$. The charge of point $i$ is $Q_i \equiv y_i\, \alpha_i$, where $\alpha_i \geq 0$ is the amount of charge, to be discussed below.

We briefly review some elementary physics. If a unit positive charge is at $\boldsymbol{x}$ in $\mathcal{X}'$, it will be attracted to all charges in $S^-$ and repelled by all charges in $S^+$. To move the charge from $\boldsymbol{x}$ to some other location $\tilde{\boldsymbol{x}}$, the attractive and repelling forces must be overcome at every point along the trajectory; the path integral of the force along the trajectory is called the *work* and does not depend on the trajectory. The *potential* at $\boldsymbol{x}$ is the work that must be done to move a unit positive charge from a reference point (usually infinity) to $\boldsymbol{x}$.

The potential at $\boldsymbol{x}$ is $\varphi(\boldsymbol{x}) = \sum_{j=1}^N Q_j\, G(\boldsymbol{x}^j, \boldsymbol{x})$, where $G$ is a function of the distance. In electrostatic systems with point charges, $G(\boldsymbol{a}, \boldsymbol{b}) = 1/\|\boldsymbol{a} - \boldsymbol{b}\|_2$. From this definition, one can see that the potential at $\boldsymbol{x}$ is negative (positive) if $\boldsymbol{x}$ is in a neighborhood of many negative (positive) charges. Thus, the potential indicates the sign and amount of charge in the local neighborhood.

Turning back to the ML classifier, one might propose a classification rule for some input $\boldsymbol{x}$ that assigns the label "+" if $\varphi(\boldsymbol{x}) > 0$ or "−" otherwise. Abstracting from the electrostatic system, if $\alpha_i = 1$ and $G$ is a function that decreases sufficiently steeply with distance, we obtain a nearest-neighbor classifier. This potential classifier can be also interpreted as Parzen windows classifier [9].

## 1.2 Electrostatic model 2: Coupled point charges

Consider now an electrostatic model that extends the previous model in two respects. First, the point charges are replaced by *conductors*, e.g., metal spheres. Each conductor $i$ has a *self–potential coefficient*, denoted $P_{ii}$, which is a measure of how much charge it can easily hold; for a metal sphere, $P_{ii}$ is related to sphere's diameter. Second, the conductors in $S^+$ are *coupled*, as are the conductors in $S^-$. "Coupling" means that charge is free to flow between the conductors. Technically, $S^+$ and $S^-$ can each be viewed as a single conductor.

In this model, we initially place the same charge $\nu/N$ on each conductor, and allow charges within $S^+$ and $S^-$ to flow freely (we assume no resistance in the coupling and no polarization of the conductors). After the charges redistribute, charge will tend to end up on the periphery of a homogeneous neighborhood of conductors, because like charges repel. Charge will also tend to end up along the $S^+$–$S^-$ boundary because opposite charges attract. Figure 1 depicts the redistribution of charges, where the shading is proportional to the magnitude $\alpha_i$. An ML classifier can be built based on this model, once again using $\varphi(\boldsymbol{x}) > 0$ as the decision rule

for classifying an input $\boldsymbol{x}$. In this model, however, the $\alpha_i$ are not uniform; the conductors with large $\alpha_i$ will have the greatest influence on the potential function. Consequently, one can think of $\alpha_i$ as the weight or importance of example $i$. As we will show shortly, the examples with $\alpha_i > 0$ are exactly support vectors of an SVM.

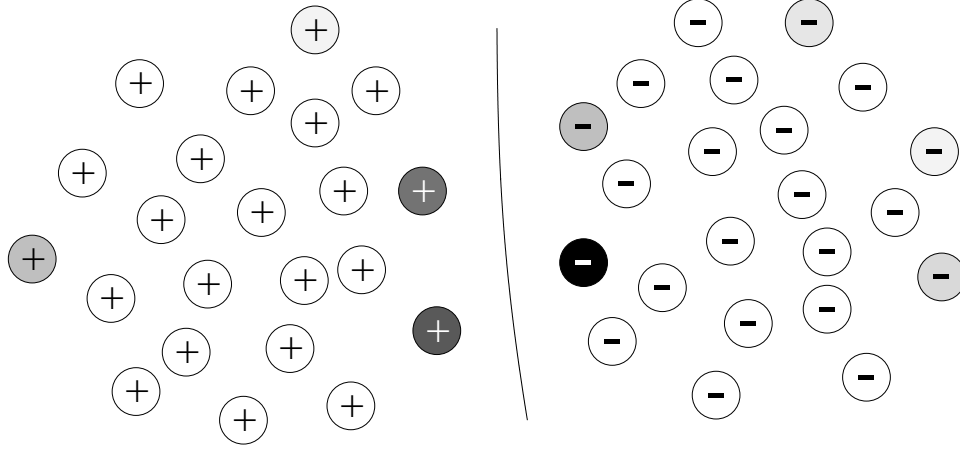

Figure 1: Coupled conductor system following charge redistribution. Shading reflects the charge magnitude, and the contour indicates a zero potential.

The redistribution of charges in the electrostatic system is achieved via minimization of the Coulomb energy. Imagine placing the same total charge magnitude, $m$, on $S^+$ and $S^-$ by dividing it uniformly among the conductors, i.e., $\alpha_i = m/\left|S^{y_i}\right|$. The free charge flow in $S^+$ and $S^-$ yields a distribution of charges, the $\alpha_i$, such that Coulomb energy is minimized.

To introduce Coulomb energy, we begin with some preliminaries. The potential at conductor $i$, $\varphi(\boldsymbol{x}^i)$, which we will denote more compactly as $\varphi_i$, can be described in terms of the *coefficients of potential* $P_{ij}$ [10]: $\varphi_i = \sum_{j=1}^{N} P_{ij}\, Q_j$, where $P_{ij}$ is the potential induced on conductor $i$ by charge $Q_j$ on conductor $j$; $P_{ii} \geq P_{ij} \geq 0$ and $P_{ij} = P_{ji}$. If each conductor $i$ is a metal sphere centered at $\boldsymbol{x}^i$ and has radius $r_i$ (radii are enforced to be small enough so that the spheres do not touch each other), the system can be modeled by a point charge $Q_i$ at $\boldsymbol{x}^i$, and $P_{ij} = G\left(\boldsymbol{x}^i, \boldsymbol{x}^j\right)$ as in the previous section [10]. The self-potential, $P_{ii}$, is defined as a function of $r_i$. The Coulomb energy is defined in terms of the potential on the conductors, $\varphi_i$:

$$E \;=\; \frac{1}{2}\sum_{i=1}^{N} \varphi_i\, Q_i \;=\; \frac{1}{2}\, \boldsymbol{Q}^T\, \boldsymbol{P}\, \boldsymbol{Q} \;=\; \frac{1}{2}\sum_{i,j=1}^{N} P_{ij}\, y_i\, y_j\, \alpha_i\, \alpha_j \;.$$

When the energy minimum is reached, the potential $\varphi_i$ will be the same for all connected $i \in S^+$ ($i \in S^-$); we denote this potential $\varphi_{S^+}$ ($\varphi_{S^-}$).

Two additional constraints on the system of coupled conductors are necessary in order to interpret the system in terms of existing machine learning models. First, the positive and negative potentials must be balanced, i.e., $\varphi_{S^+} = -\varphi_{S^-}$. This constraint is achieved by setting the reference point of the potentials through $b$, $b = -0.5\left(\varphi_{S^+} + \varphi_{S^-}\right)$, into the potential function: $\varphi\left(\boldsymbol{x}\right) = \sum_{i=1}^{N} Q_i\, G\left(\boldsymbol{x}^i, \boldsymbol{x}\right) + b$. Second, the conductors must be prevented from reversing the sign of their charge, i.e., $\alpha_i \geq 0$, and from holding more than a quantity $C$ of charge, i.e., $\alpha_i \leq C$. These

requirements can be satisfied in the electrostatic model by disconnecting a conductor $i$ from the charge flow in $S^+$ or $S^-$ when $\alpha_i$ reaches a bound, which will subsequently freeze its charge. Mathematically, the requirements are satisfied by treating energy minimization as a constrained optimization problem with $0 \leq \alpha_i \leq C$.

The electrostatic system corresponds to a $\nu$–support vector machine ($\nu$–SVM) [9] with kernel $G$ if we set $C = 1/N$. The electrostatic system assures that $\sum_{i \in S^+} \alpha_i = \sum_{i \in S^-} \alpha_i = 0.5 \ \nu$. The identity holds because the Coulomb energy is exactly the $\nu$–SVM quadratic objective function, and the thresholded electrostatic potential evaluated at a location is exactly the SVM decision rule. The minimization of potentials differences in the systems $S^+$ and $S^-$ corresponds to the minimization of slack variables in the SVM (slack variables express missing potential due to the upper bound on $\alpha_i$). Mercer's condition [6], the essence of the nonlinear SVM theory, is equivalent to the fact that continuous electrostatic energy is positive, i.e., $E = \int G(\boldsymbol{x}, \boldsymbol{z}) \ h(\boldsymbol{x}) \ h(\boldsymbol{z}) \ d\boldsymbol{x} \ d\boldsymbol{z} \geq 0$. The self-potentials of the electrostatic system provide an interpretation to the diagonal elements in the quadratic objective function of the SVM. This interpretation of the diagonal elements allows us to introduce novel kernels and novel SVM methods, as we discuss later.

### 1.3 Electrostatic model 3: Coupled point charges with battery

In electrostatic model 2, we control the magnitude of charge applied to $S^+$ and $S^-$. Although we apply the same charge magnitude to each, we do not have to control the resulting potentials $\varphi_{S+}$ and $\varphi_{S-}$, which may be imbalanced. We compensate for this imbalance via the potential offset $b$. In electrostatic model 3, we control the potentials $\varphi_{S+}$ and $\varphi_{S+}$ directly by adding a battery to the system. We connect $S^+$ to the positive pole of the battery with potential $+1$ and $S^-$ to the negative pole with potential $-1$. The battery ensures that $\varphi_{S+} = +1$ and $\varphi_{S-} = -1$ because charges flow from the battery into or out of the system until the systems take on the potential of the battery poles. The battery can then be removed. The potential $\varphi_i = y_i$ is forced by the battery on conductor $i$. The total Coulomb energy is the energy from model 2 minus the work done by the battery. The work done by the battery is $\sum_{i \leq N} y_i Q_i = \sum_{i \leq N} \alpha_i$. The Coulomb energy is

$$\frac{1}{2} \ \boldsymbol{Q}^T \ \boldsymbol{P} \ \boldsymbol{Q} \ - \ \sum_{i=1}^{N} \alpha_i \ = \frac{1}{2} \sum_{i,j=1}^{N} P_{ij} \ y_i \ y_j \ \alpha_i \ \alpha_j \ - \ \sum_{i=1}^{N} \alpha_i \ .$$

This physical system corresponds to a $C$–support vector machine ($C$–SVM) [2, 11]. The $C$–SVM requires that $\sum_i y_i \alpha_i = 0$; although this constraint may not be fulfilled in the system described here, it can be enforced by a slightly different system [4]. A more straightforward relation to the $C$–SVM is given in [9] where the authors show that every $\nu$–SVM has the same class boundaries as a $C$–SVM with appropriate $C$.

## 2  Comparison of existing and novel models

### 2.1 Novel Kernels

The electrostatic perspective makes it easy to understand why SVM algorithms can break down in high-dimensional spaces: Kernels with rapid fall-off induce small potentials and consequently, almost every conductor retains charge. Because a charged conductor corresponds to a support vector, the number of support vectors is large, which leads to two disadvantages: (1) the classification procedure is slow, and (2) the expected generalization error increases with the number of support vectors [11]. We therefore should use kernels that do not drop off exponentially. The self–potential

permits the use of kernels that would otherwise be invalid, such as a generalization of the electric field: $G\left(\boldsymbol{x}^i, \boldsymbol{x}^j\right) := \left\|\boldsymbol{x}^i - \boldsymbol{x}^j\right\|_2^{-l}$ and $G\left(\boldsymbol{x}^i, \boldsymbol{x}^i\right) := r_i^{-l} = P_{ii}$, where $r_i$ the radius of the $i$th sphere. The $r_i$s are increased to their maximal values, i.e. until they hit other conductors $(r_i = 0.5\min_j \left\|\boldsymbol{x}^i - \boldsymbol{x}^j\right\|_2)$. These kernels, called "Coulomb kernels", are invariant to scaling of the input space in the sense that scaling does not change the minimum of the objective function. Consequently, such kernels are appropriate for input data with varying local densities. Figure 2 depicts a classification task with input regions of varying density. The optimal class boundary is smooth in the low data density regions and has high curvature in regions, where the data density is high. The classification boundary was constructed using a $C$-SVM with a Plummer kernel $G\left(\boldsymbol{x}^i, \boldsymbol{x}^j\right) := \left(\left\|\boldsymbol{x}^i - \boldsymbol{x}^j\right\|_2^2 + \epsilon^2\right)^{-l/2}$, which is an approximation to our novel Coulomb kernel but lacks its weak singularities.

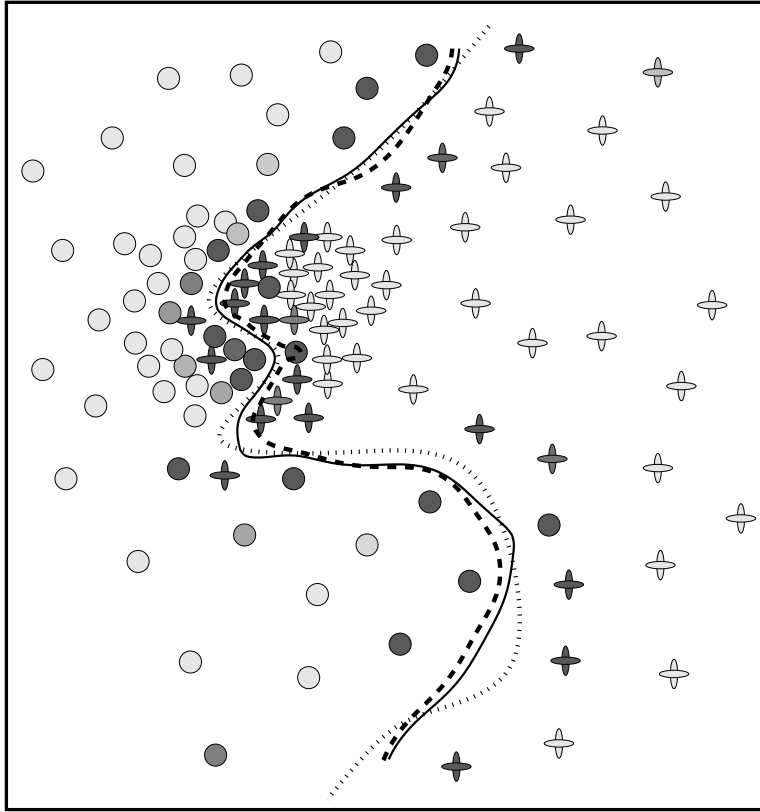

Figure 2: Two class data with a dense region and trained with a SVM using the new kernel. Gray-scales indicate the weights — support vectors are dark. Boundary curves are given for the novel kernel (solid), best RBF-kernel SVM which overfits at high density regions where the resulting boundary goes through a dark circle (dashed), and optimal boundary (dotted).

### 2.2 Novel SVM models

Our electrostatic framework can be used to derive novel SVM approaches [4], two representative examples of which we illustrate here.

*2.2.1 $\kappa$–Support Vector Machine ($\kappa$–SVM):*
We can exploit the physical interpretation of $P_{ii}$ as conductor $i$'s self–potential. The $P_{ii}$'s determine the smoothness of the charge distribution at the energy minimum. We can introduce a parameter $\kappa$ to rescale the self potential – $P_{ii}^{new} = \kappa\ P_{ii}^{old}$. $\kappa$ controls the complexity of the corresponding SVM. With this modification, and with $C = \infty$, electrostatic model 3 becomes what we call the $\kappa$–SVM.

*2.2.2 p–Support Vector Machine (p–SVM):*
At the Coulomb energy minimum the electrostatic potentials equalize: $\varphi_i - y_i = 0$, $\forall i$ ($\boldsymbol{y}$ is the label vector). This motivates the introduction of potential difference, $\frac{1}{2}\|PQ + \boldsymbol{y}\|_2^2 = \frac{1}{2}\boldsymbol{Q}^T\boldsymbol{P}^T\boldsymbol{P}\boldsymbol{Q} + \boldsymbol{Q}^T\boldsymbol{P}^T\boldsymbol{y} + \frac{1}{2}\boldsymbol{y}^T\boldsymbol{y}$ as the objective. We obtain

$$\min_{\boldsymbol{\alpha}} \quad \frac{1}{2}\boldsymbol{\alpha}^T\ \boldsymbol{Y}\ \boldsymbol{P}^T\ \boldsymbol{P}\ \boldsymbol{Y}\ \boldsymbol{\alpha}\ -\ \boldsymbol{1}^T\boldsymbol{Y}\ \boldsymbol{P}\ \boldsymbol{Y}\ \boldsymbol{\alpha}$$

$$\text{subject to} \quad \boldsymbol{1}^T\ \boldsymbol{P}\ \boldsymbol{Y}\boldsymbol{\alpha}\ =\ 0\ ,\ |\alpha_i|\ \leq\ C,$$

where $\boldsymbol{1}$ is the vector of ones and $\boldsymbol{Y} := \text{diag}(\boldsymbol{y})$. We call this variant of the optimization problem the potential-SVM (p-SVM). Note that the p-SVM is similar to the "empirical kernel map" [9]. However $\boldsymbol{P}$ appears in the objective's linear term and the constraints. We classify in a space where $\boldsymbol{P}$ is a dot product matrix. The constraint $\boldsymbol{1}^T\ \boldsymbol{P}\ \boldsymbol{Y}\boldsymbol{\alpha}\ =\ 0$ ensures that the average potential for each class is equal.

By construction, $\boldsymbol{P}^T\ \boldsymbol{P}$ is positive definite; consequently, *this formulation does not require positive definite kernels.* This characteristic is useful for problems in which the properties of the objects to be classified are described by their pairwise proximities. That is, suppose that instead of representing each input object by an explicit feature vector, the objects are represented by a matrix which contains a real number indicating the similarity of each object to each other object. We can interpret the entries of the matrix as being produced by an unknown kernel operating on unknown feature vectors. In such a matrix, however, positive definiteness cannot be assured, and the optimal hyperplane must be constructed in Minkowski space.

## 3 Experiments

**UCI Benchmark Repository.** For the representative models we have introduced, we perform simulations and make comparisons to standard SVM variants. All datasets (except "banana" from [7]) are from the UCI Benchmark Repository and were preprocessed in [7]. We did 100-fold validation on each data set, restricting the training set to 200 examples, and using the remainder of examples for testing. We compared two standard architectures, the $C$–SVM and the $\nu$–SVM, to our novel architectures: to the $\kappa$–SVM, to the p–SVM, and to a combination of them, the $\kappa$–p–SVM. The $\kappa$–p–SVM is a p–SVM regularized like a $\kappa$–SVM. We explored the use of radial basis function (RBF), polynomial (POL), and Plummer (PLU) kernels. Hyperparameters were determined by 5–fold cross validation on the first 5 training sets. The search for hyperparameter was not as intensive as in [7].

Table 1 shows the results of our comparisons on the UCI Benchmarks. Our two novel architectures, the $\kappa$–SVM and the p–SVM, performed well against the two existing architectures (note that the differences between the $C$– and the $\nu$–SVM are due to model selection). As anticipated, the p–SVM requires far fewer support vectors. Additionally, the Plummer kernel appears to be more robust against hyperparameter and SVM choices than the RBF or polynomial kernels.

|       | $C$ | $\nu$ | $\kappa$ | p | $\kappa$-p | $C$ | $\nu$ | $\kappa$ | p | $\kappa$-p |
|-------|-----|-------|----------|---|-----------|-----|-------|----------|---|-----------|
|       | thyroid | | | | | heart | | | | |
| RBF | 6.4 | 9.4 | 7.7 | **5.4** | 8.6 | 21.4 | 19.1 | 17.9 | 22.4 | *17.8* |
| POL | 22.8 | 12.6 | 7.0 | 13.3 | 6.9 | 20.4 | 20.4 | 19.3 | 23.0 | 19.3 |
| PLU | *6.1* | 6.2 | *6.1* | *5.7* | *6.1* | **16.3** | **16.3** | **16.3** | *17.4* | **16.3** |
|       | breast–cancer | | | | | banana | | | | |
| RBF | 33.6 | 31.6 | 33.8 | 32.4 | 33.7 | *13.2* | 36.7 | *13.2* | *11.6* | 13.4 |
| POL | 36.0 | **25.7** | 29.6 | *27.1* | *29.1* | 35.3 | 35.0 | **11.5** | 22.4 | **11.5** |
| PLU | 33.4 | 33.1 | 33.4 | 30.6 | 33.4 | 15.7 | 15.7 | 15.7 | 21.9 | 15.7 |
|       | german | | | | | | | | | |
| RBF | 28.7 | 29.3 | 29.0 | *27.8* | 28.8 | | | | | |
| POL | 33.7 | 29.6 | **26.2** | 31.8 | **26.2** | | | | | |
| PLU | 28.8 | 28.5 | 33.3 | *27.1* | 33.3 | | | | | |

Table 1: Mean % misclassification on 5 UCI Repository data sets. Each cell in the table is obtained via 100 replications splitting the data into training and test sets. The comparison is among five SVMs (the table columns) using three kernel functions (the table rows). Cells in bold face are the best result for a given data set and italicized the second and third best.

**Pairwise Proximity Data.** We applied our p–SVM and the generalized SVM (G–SVM) [3] to two pairwise-proximity data sets. The first data set, the "cat cortex" data, is a matrix of connection strengths between 65 cat cortical areas and was provided by [8], where the available anatomical literature was used to determine proximity values between cortical areas. These areas belong to four different coarse brain regions: auditory (A), visual (V), somatosensory (SS), and frontolimbic (FL). The goal was to classify a given cortical area as belonging to a given region or not. The second data set, the "protein" data, is the evolutionary distance of 226 sequences of amino acids of proteins obtained by a structural comparison [5] (provided by M. Vingron). Most of the proteins are from four classes of globins: hemoglobin-ff (H-ff), hemoglobin-fi (H-fi), myoglobin (M), and heterogenous globins (GH). The goal was to classify a protein as belonging to a given globin class or not. As Table 2 shows, our novel architecture, the p–SVM, beats out an existing architecture in the literature, the G–SVM, on 5 of 8 classification tasks, and ties the G–SVM on 2 of 8; it loses out on only 1 of 8.

|        | cat cortex | | | | | protein data | | | | |
|--------|------|-----|-----|-----|-----|------|-------------|------------|-----|-----|
|        | Reg. | V | A | SS | FL | Reg. | H-$\alpha$ | H-$\beta$ | M | GH |
| Size   | — | 18 | 10 | 18 | 19 | — | 72 | 72 | 39 | 30 |
| G-SVM | 0.05 | 4.6 | 3.1 | **3.1** | **1.5** | 0.05 | 1.3 | 4.0 | 0.5 | 0.5 |
| G-SVM | 0.1 | 4.6 | 3.1 | 6.1 | **1.5** | 0.1 | 1.8 | 4.5 | 0.5 | 0.9 |
| G-SVM | 0.2 | 6.1 | **1.5** | **3.1** | 3.1 | 0.2 | 2.2 | 8.9 | 0.5 | 0.9 |
| p-SVM | 0.6 | **3.1** | **1.5** | 6.1 | 3.1 | 300 | **0.4** | 3.5 | **0.0** | **0.4** |
| p-SVM | 0.7 | **3.1** | 3.1 | 4.6 | **1.5** | 400 | **0.4** | **3.1** | **0.0** | 0.9 |
| p-SVM | 0.8 | **3.1** | 3.1 | 4.6 | **1.5** | 500 | **0.4** | 3.5 | **0.0** | 1.3 |

Table 2: Mean % misclassifications for the cat-cortex and protein data sets using the p–SVM and the G–SVM and a range of regularization parameters (indicated in the column labeled "Reg."). The result obtained for the cat-cortex data is via leave-one-out cross validation, and for the protein data is via ten-fold cross validation. The best result for a given classification problem is printed in bold face.

## 4 Conclusion

The electrostatic framework and its analogy to SVMs has led to several important ideas. First, it suggests SVM methods for kernels that are not positive definite. Second, it suggests novel approaches and kernels that perform as well as standard methods (will undoubtably perform better on some problems). Third, we demonstrated a new classification technique working in Minkowski space which can be used for data in form of pairwise proximities. The novel approach treats the proximity matrix as an SVM Gram matrix which lead to excellent experimental results.

We argued that the electrostatic framework not only characterizes a family of support-vector machines, but it also characterizes other techniques such as nearest neighbor classification. Perhaps the most important contribution of the electrostatic framework is that, by interrelating and encompassing a variety of methods, it lays out a broad space of possible algorithms. At present, the space is sparsely populated and has barely been explored. But by making the dimensions of this space explicit, the electrostatic framework allows one to easily explore the space and discover novel algorithms. In the history of machine learning, such general frameworks have led to important advances in the field.

**Acknowledgments**
We thank G. Hinton and J. Schmidhuber for stimulating conversations leading to this research and an anonymous reviewer who provided helpful advice on the paper.

## References

[1] M. A. Aizerman, E. M. Braverman, and L. I. Rozonoér. Theoretical foundations of the potential function method in pattern recognition learning. *Automation and Remote Control*, 25:821–837, 1964.

[2] C. J. C. Burges. A tutorial on support vector machines for pattern recognition. *Data Mining and Knowledge Discovery*, 2(2):1–47, 1998.

[3] T. Graepel, R. Herbrich, B. Schölkopf, A. J. Smola, P. L. Bartlett, K.-R. Müller, K. Obermayer, and R. C. Williamson. Classification on proximity data with LP–machines. In *Proceedings of the Ninth International Conference on Artificial Neural Networks*, pages 304–309, 1999.

[4] S. Hochreiter and M. C. Mozer. Coulomb classifiers: Reinterpreting SVMs as electrostatic systems. Technical Report CU-CS-921-01, Department of Computer Science, University of Colorado, Boulder, 2001.

[5] T. Hofmann and J. Buhmann. Pairwise data clustering by deterministic annealing. *IEEE Trans. Pattern Anal. and Mach. Intelligence*, 19(1):1–14, 1997.

[6] J. Mercer. Functions of positive and negative type and their connection with the theory of integral equations. *Philosophical Transactions of the Royal Society of London* **A**, 209:415–446, 1909.

[7] G. Rätsch, T. Onoda, and K.-R. Müller. Soft margins for AdaBoost. Technical Report NC-TR-1998-021, Dep. of Comp. Science, Univ. of London, 1998.

[8] J. W. Scannell, C. Blakemore, and M. P. Young. Analysis of connectivity in the cat cerebral cortex. *The Journal of Neuroscience*, 15(2):1463–1483, 1995.

[9] B. Schölkopf and A. J. Smola. *Learning with Kernels — Support Vector Machines, Regularization, Optimization, and Beyond.* MIT Press, 2002.

[10] M. Schwartz. *Principles of Electrodynamics.* Dover Publications, NY, 1987. Republication of McGraw-Hill Book 1972.

[11] V. Vapnik. *The nature of statistical learning theory.* Springer, NY, 1995.
